# Data Analysis using G/SPLINES

**David Rogers** [*]
Research Institute for Advanced Computer Science
MS T041-5, NASA/Ames Research Center
Moffett Field, CA 94035
INTERNET: drogers@riacs.edu

## Abstract

G/SPLINES is an algorithm for building functional models of data. It uses genetic search to discover combinations of basis functions which are then used to build a least-squares regression model. Because it produces a population of models which evolve over time rather than a single model, it allows analysis not possible with other regression-based approaches.

## 1 INTRODUCTION

G/SPLINES is a hybrid of Friedman's Multivariable Adaptive Regression Splines (MARS) algorithm (Friedman, 1990) with Holland's Genetic Algorithm (Holland, 1975).

G/SPLINES has advantages over MARS in that it requires fewer least-squares computations, is easily extendable to non-spline basis functions, may discover models inaccessible to local-variable selection algorithms, and allows significantly larger problems to be considered. These issues are discussed in (Rogers, 1991).

This paper begins with a discussion of linear regression models, followed by a description of the G/SPLINES algorithm, and finishes with a series of experiments illustrating its performance, robustness, and analysis capabilities.

* Currently at Polygen/Molecular Simulations, Inc., 796 N. Pastoria Ave., Sunnyvale, CA 94086, INTERNET: drogers@msi.com.

## 2 LINEAR MODELS

A common assumption used in data modeling is that the data samples are derived from an underlying function:

$$y_i = f(X_i) + \text{error}$$
$$\text{""} = f(x_{i1}, ..., x_{in}) + \text{error}$$

The goal of analysis is to develop a model $F(X)$ which minimizes the least-squares error:

$$LSE(F) = \frac{1}{N} \sum_{i=1}^{N} (y_i - F(X_i))^2$$

The function $F(X)$ can then be used to estimate the underlying function f at previously-seen data samples (*recall*) or at new data samples (*prediction*). Samples used to construct the function $F(X)$ are in the *training set*; samples used to test prediction are in the *test set*.

In constructing $F(X)$, if we assume the model F can be written as a linear combination of basis function $\{\phi_\kappa\}$:

$$F(X) = a_0 + \sum_{k=1}^{M} a_k \phi_k(X)$$

then standard least-squares regression can find the optimal coefficients $\{a_k\}$. However, selecting an appropriate set of basis functions for high-dimensional models can be difficult. G/SPLINES is a primarily a method for selecting this set.

## 3 G/SPLINES

Many techniques develop a regression model by incremental addition or deletion of basis functions to a single model.The primary idea of G/SPLINES is to keep a *collection* of models, and use the genetic algorithm to recombine among these models.

G/SPLINES begins with a collection of models containing randomly-generated basis functions.

$$F_1: \{\phi_1 \ \phi_2 \ \phi_3 \ \phi_4 \ \phi_5 \ \phi_6 \ \phi_7 \ \phi_8 \ \phi_9 \ \phi_{10} \ \phi_{11} \ \phi_{12} \ \phi_{13} \ \phi_{14}\}$$
$$F_2: \{\delta_1 \ \delta_2 \ \delta_3 \ \delta_4 \ \delta_5 \ \delta_6 \ \delta_7 \ \delta_8 \ \delta_9 \ \delta_{10} \ \delta_{11}\}$$
$$\vdots \qquad\qquad \vdots$$
$$F_K: \{\sigma_1 \ \sigma_2 \ \sigma_3 \ \sigma_4 \ \sigma_5 \ \sigma_6 \ \sigma_7 \ \sigma_8 \ \sigma_9 \ \sigma_{10} \ \sigma_{11} \ \sigma_{12}\}$$

The basis functions are functions which use a small number of the variables in the data set, such as $SIN(x_2 - 1)$ or $(x_4 - .4)(x_5 - .1)$. The model coefficients $\{a_k\}$ are determined using least-squares regression.

Each model is scored using Friedman's "lack of fit" (LOF) measure, which is a penalized least-squares measure for goodness of fit; this measure takes into account factors such as the number of data samples, the least-squares error, and the number of model parameters.

At this point, we repeatedly perform the *genetic crossover* operation:

- Two good models are probabilistically selected as "parents". The likelihood of being chosen is inversely proportional to a model's LOF score.

- Each parent is randomly "cut" into two sections, and a new model is created using a piece from each parent:

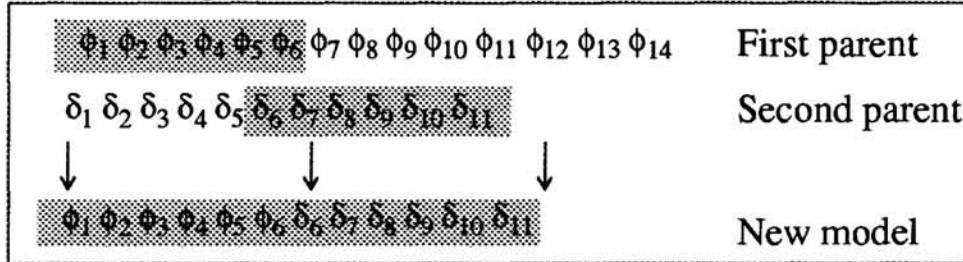

- Optional mutation operators may alter the newly-created model.

- The model with the worst LOF score is replaced by this new model.

This process ends when the average fitness of the population stops improving.

Some features of the G/SPLINES algorithm are significantly different from MARS:

Unlike incremental search, full-sized models are tested at every step.
The algorithm automatically determines the proper size for models.
Many fewer models are tested than with MARS.
A population of models offers information not available from single-model methods.

## 4 MUTATION OPERATORS

Additional mutation operators were added to the system to counteract some negative tendencies of a purely crossover-based algorithm.

**Problem**: genetic diversity is reduced as process proceeds (fewer basis functions in population)

NEW: creates a new basis function by randomly choosing a basis function type and then randomly filling in the parameters.

**Problem**: need process for constructing useful multidimensional basis functions

MERGE: takes a random basis function from each parent, and creates a new basis function by multiplying them together.

**Problem**: models contain "hitchhiking" basis functions which contribute little

DELETION: ranks the basis functions in order of minimum maximum contribution to the approximation. It removes one or more of the least-contributing basis functions.

## 5 EXPERIMENTAL

Experiments were conducted on data derived from a function used by Friedman (1988):

$$f(X) = SIN(\pi X_1 X_2) + 20(X_3 - \frac{1}{2})^2 + 10X_4 + 5X_5$$

Standard experimental conditions are as follows. Experiments used a training set containing 200 samples, and a test set containing 200 samples. Each sample contained 10 predictor variables (5 informative, 5 noninformative) and a response. Sample points were randomly selected from within the unit hypercube. The signal/noise ratio was 4.8/1.0

The G/SPLINE population consisted of 100 models. Linear truncated-power splines were used as basis functions. After each crossover, a model had a 50% chance of getting a new basis function created by operator NEW or MERGE and the least-contributing 10% of its basis functions deleted using operator DELETE.

The standard training phase involved 10,000 crossover operations. After training, the models were tested against a set of 200 previously-unseen test samples.

## 5.1 G/SPLINES VS. MARS

**Question**: is G/SPLINE competitive with MARS?

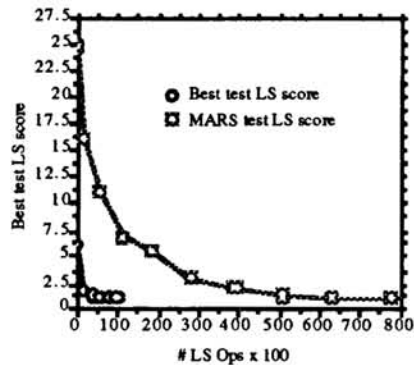

Figure 1. Test least-squares scores versus number of least-squares regressions for G/SPLINES and MARS.

The MARS algorithm was close to convergence after 50,000 least-squares regressions, and showed no further improvement after 80,000. The G/SPLINES algorithm was close to convergence after 4,000 least-squared regressions, and showed no further improvement after 10,000. [Note: the number of least-squares regressions is not a direct measure of the computational efficiency of the algorithms, as MARS uses a technique (applicable only to linear truncated-power splines) to greatly reduce cost of doing least-squares-regression.]

To complete the comparison, we need results on the quality of the discovered models:

Final average least-squared error of the best 4 G/SPLINES models was:     ~1.17
Final least-squared error of the MARS model was:     ~1.12
The "best" model has a least-squared error (from the added noise) of:     ~1.08

Using only linear truncated-power splines, G/SPLINES builds models comparable (though slightly inferior) to MARS. However, by using basis functions other than linear truncated power splines, G/SPLINES can build improved models. If we repeat the experiment with additional basis function types of step functions, linear splines, and quadratic splines, we get improved results:

With additional basis functions, the final average least-squared error was:     ~1.095.

I suggest that by including basis functions which reflect the underlying structure of f, the quality of the discovered models is improved.

## 5.2 VARIABLE ELIMINATION

**Question**: does variable usage in the population reflect the underlying function? (Recall that the data samples contained 10 variables; only the first 5 were used to calculate f.)

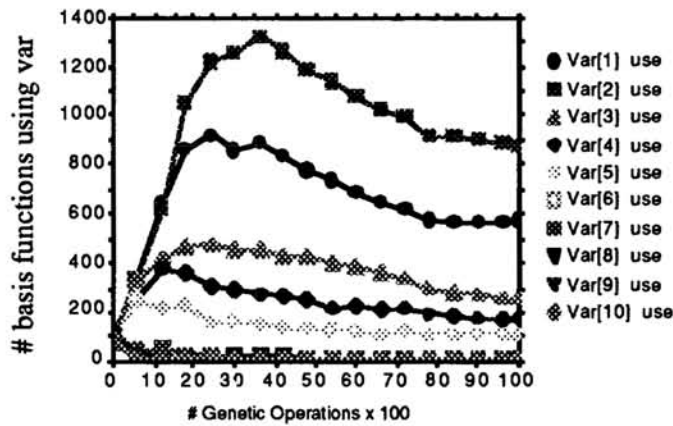

Figure 2. # of basis functions using a variable vs. # of crossover operations.

G/SPLINES correctly focuses on basis functions which use the first five variables The relative usage of these five variables reflects the complexity of the relationship between an input variable and the response in a given dimension.

**Question**: is the rate of elimination of variables affected by sample size?

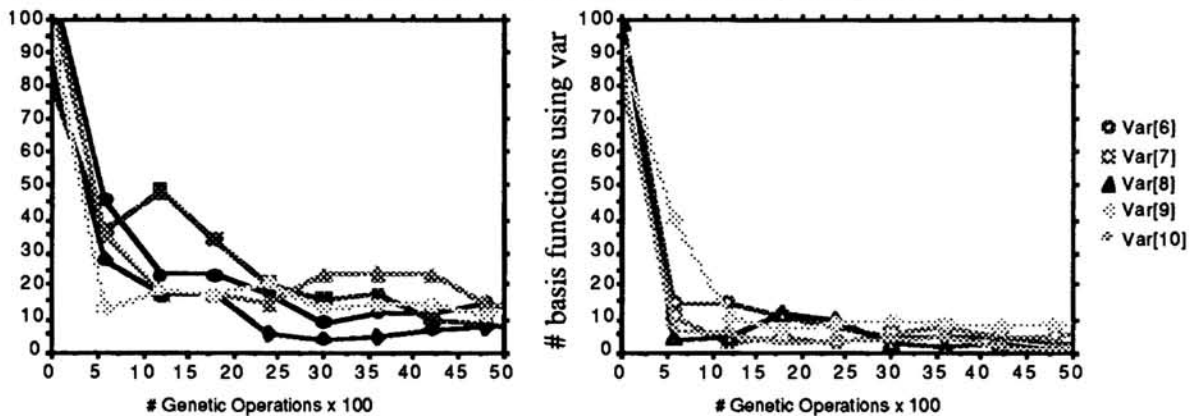

Figure 3. Close-up of Figure 2, showing the five variables not affecting the response. The left graph is the standard experiment; the right from a training with 50 samples.

The left graph plots the number of basis functions containing a variable versus the number of genetic operations for the five noninformative variables in the standard experiment. The variables are slowly eliminated from consideration. The right graph plots the same information, using a training set size of 50 samples. The variables are rapidly eliminated. Smaller training sets force the algorithm to work with most predictive variables, causing a faster elimination of less predictive variables.

**Question**: Is variable elimination effective with increased numbers of noninformative variables?

This experiment used the standard conditions but increased the number of predictor variables in the training and test sets to 100 (5 informative, 95 noninformative).

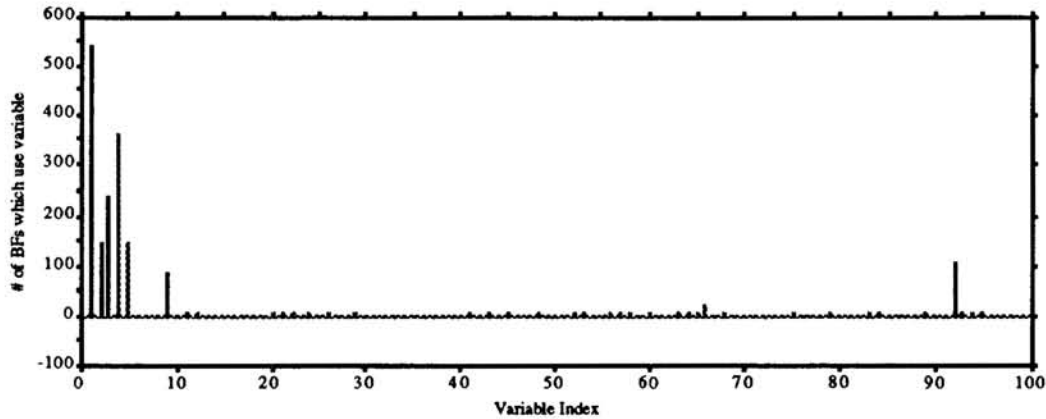

Figure 4. Number of basis functions which used a variable vs. variable index, after 10,000 genetic operations.

Figure 4 shows that elimination behavior was still apparent in this high-dimensional data set. The five informative variables were the first five in order of use.

## 5.3 MODEL SIZE

**Question:** What is the effect of the genetic algorithm on model size?

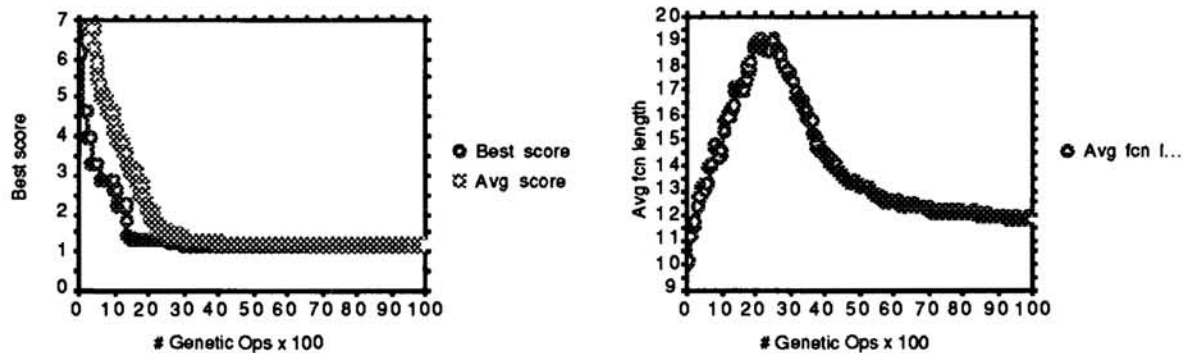

Figure 5. Model scores on training set and average function length.

The left graph plots the best and average LOF score for the training set versus the number of genetic operations. The right graph plots the average number of basis functions in a model versus the number of genetic operations.

Even after the LOF error is minimized, the average model length continues to decrease. This is likely due to pressure from the genetic algorithm; a compact representation is more likely to survive the crossover operation without loss. (In fact, due to the nature of the LOF function, the least-squared errors of the best models is slightly increased by this procedure. The system considers the increase a fair trade-off for smaller model size.)

## 5.4 RESISTANCE TO OVERFITTING

**Question:** Does Friedman's LOF function resist overfitting with small training sets?

Training was conducted with data sets of two sizes: 200 and 50. The left graph in Figure 6 plots the population average least-squared error for the training set and the test set versus the number of genetic operations, using a training set size of 200 samples. The right graph

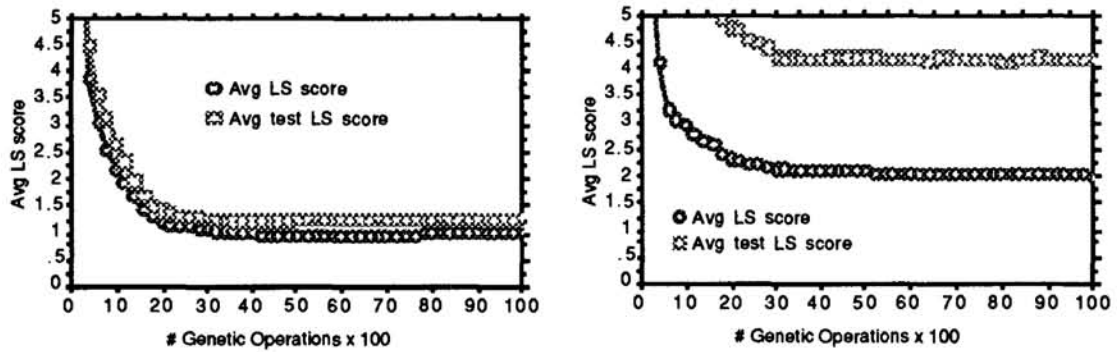

Figure 6. LS error vs. # of operations for training with 200 and 50 samples.

plots the same information, but for a system using a training set size of 50 samples.

In both cases, little overfitting is seen, even when the algorithm is allowed to run long after the point where improvement ceases. Training with a small number of samples still leads to models which resist overfitting.

**Question**: What is the effect of additive noise on overfitting?

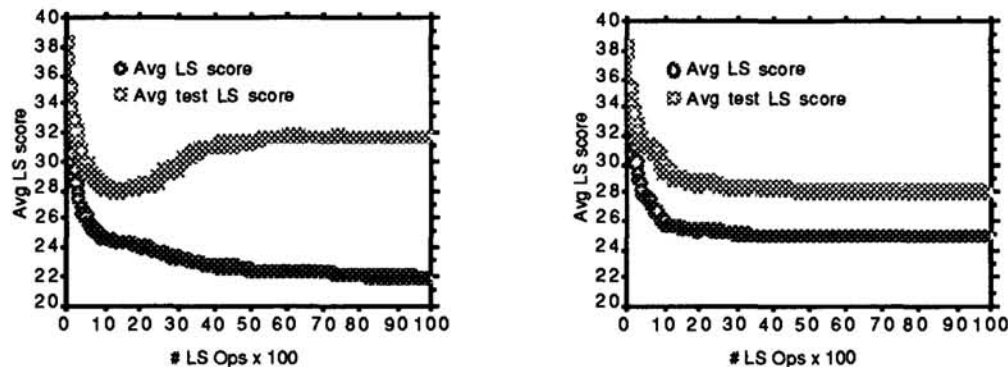

Figure 7. LS error vs. # of operations for low and high noise data sets.

Training was conducted with training sets having a signal/noise ratio of 1.0/1.0. The left graph plots the least-squared error for the training and test set versus the number of genetic operations. The right graph plots the same information, but with a higher setting of Friedman's smoothing parameter.

Noisy data results in a higher risk of overfitting. However, this can be accommodated if we set a higher value for Friedman's smoothing parameter.

## 5.5 ADDITIONAL BASIS FUNCTION TYPES AND TRAINING SET SIZES

**Question**: What is the effect of changes in training set size on the type of basis functions selected?

The experiment in Figure 8 used the standard conditions, but using many additional basis function types. The left graph plots the use of different types of basis functions using a training set of size 50.The right graph plots the same information using a training set size of 200. Simply put, different training set sizes lead to significant changes in preferences among function types. A detailed analysis of these graphs can give insight into the nature of the data and the best components for model construction.

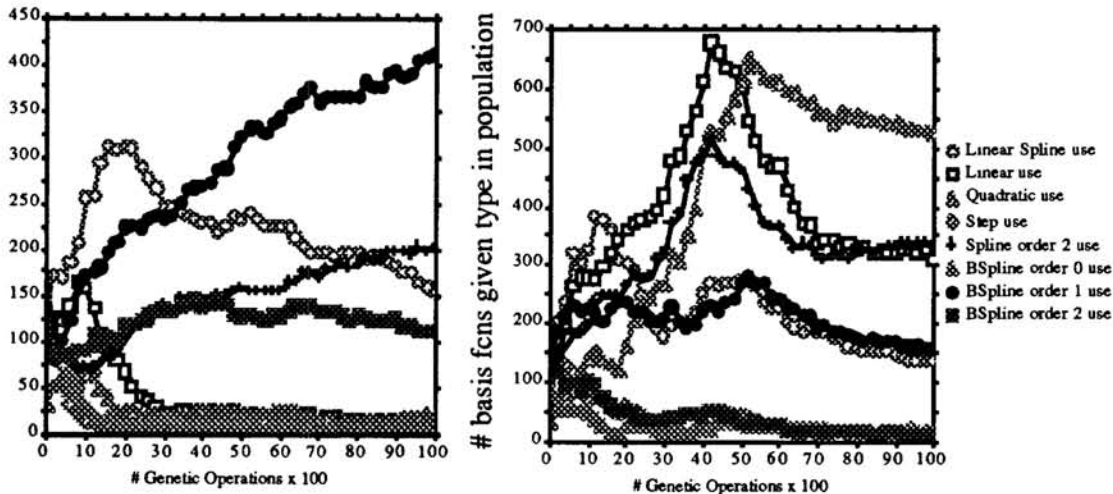

Figure 8. # of basis functions of a given type vs. # of genetic operations, for training sets of 50 and 200 samples.

# 6 CONCLUSIONS

G/SPLINES is a new algorithm related to state-of-the-art statistical modeling techniques such as MARS. The strengths of this algorithm are that G/SPLINES builds models that are comparable in quality to MARS, with a greatly reduced number of intermediate model constructions; is capable of building models from data sets that are too large for the MARS algorithm; and is easily extendable to basis functions that are not spline-based.

Weaknesses of this algorithm include the ad-hoc nature of the mutation operators; the lack of studies of the real-time performance of G/SPLINES vs. other model builders such as MARS; the need for theoretical analysis of the algorithm's convergence behavior; the LOF function needs to be changed to reflect additional basis function types.

The WOLF program source code, which implements G/SPLINES, is available free to other researchers in either Macintosh or UNIX/C formats. Contact the author (drogers@riacs.edu) for information.

### Acknowledgments

This work was supported in part by Cooperative Agreements NCC 2-387 and NCC 2-408 between the National Aeronautics and Space Administration (NASA) and the Universities Space Research Association (USRA). Special thanks to my domestic partner Doug Brockman, who shared my enthusiasm even though he didn't know what the hell I was up to; and my father, Philip, who made me want to become a scientist.

### References

Friedman, J., "Multivariate Adaptive Regression Splines," Technical Report No. 102, Laboratory for Computational Statistics, Department of Statistics, Stanford University, November 1988 (revised August 1990).

Holland, J., *Adaptation in Artificial and Natural Systems*, University of Michigan Press, Ann Arbor, MI, 1975.

Rogers, David, "G/SPLINES: A Hybrid of Friedman's Multivariate Adaptive Splines (MARS) Algorithm with Holland's Genetic Algorithm," in *Proceedings of the Fourth International Conference on Genetic Algorithms*, San Diego, July, 1991.